# Distributed Optimization in Adaptive Networks

**Ciamac C. Moallemi**
Electrical Engineering
Stanford University
Stanford, CA 94305
ciamac@stanford.edu

**Benjamin Van Roy**
Management Science and Engineering
and Electrical Engineering
Stanford University
Stanford, CA 94305
bvr@stanford.edu

## Abstract

We develop a protocol for optimizing dynamic behavior of a network of simple electronic components, such as a sensor network, an ad hoc network of mobile devices, or a network of communication switches. This protocol requires only local communication and simple computations which are distributed among devices. The protocol is scalable to large networks. As a motivating example, we discuss a problem involving optimization of power consumption, delay, and buffer overflow in a sensor network.

Our approach builds on policy gradient methods for optimization of Markov decision processes. The protocol can be viewed as an extension of policy gradient methods to a context involving a team of agents optimizing aggregate performance through asynchronous distributed communication and computation. We establish that the dynamics of the protocol approximate the solution to an ordinary differential equation that follows the gradient of the performance objective.

## 1  Introduction

This paper is motivated by the potential of policy gradient methods as a general approach to designing simple scalable distributed optimization protocols for networks of electronic devices. We offer a general framework for such protocols that builds on ideas from the policy gradient literature. We also explore a specific example involving a network of sensors that aggregates data. In this context, we propose a distributed optimization protocol that minimizes power consumption, delay, and buffer overflow.

The proposed approach for designing protocols based on policy gradient methods comprises one contribution of this paper. In addition, this paper offers fundamental contributions to the policy gradient literature. In particular, the kind of protocol we propose can be viewed as extending policy gradient methods to a context involving a team of agents optimizing system behavior through asynchronous distributed computation and parsimonious local communication. Our main theoretical contribution is to show that the dynamics of our protocol approximate the solution to an ordinary differential equation that follows the gradient of the performance objective.

## 2 A General Formulation

Consider a network consisting of a set of components $V = \{1, \ldots, n\}$. Associated with this network is a discrete-time dynamical system with a finite state space $\mathbb{W}$. Denote the state of the system at time $k$ by $w(k)$, for $k = 0, 1, 2, \ldots$. There are $n$ subsets $\mathbb{W}_1, \ldots, \mathbb{W}_n$ of $\mathbb{W}$, each consisting of states associated with events at component $i$. Note that these subsets need not be mutually exclusive or totally exhaustive. At the $k$th epoch, there are $n$ control actions $a_1(k) \in \mathbb{A}_1, \ldots, a_n(k) \in \mathbb{A}_n$, where each $\mathbb{A}_i$ is a finite set of possible actions that can be taken by component $i$. We sometimes write these control actions in a vector form $a(k) \in \mathbb{A} = \mathbb{A}_1 \times \cdots \mathbb{A}_n$. The actions are governed by a set of policies $\pi^1_{\theta_1}, \ldots, \pi^n_{\theta_n}$, parameterized by vectors $\theta_1 \in \mathbb{R}^{N_1}, \ldots, \theta_n \in \mathbb{R}^{N_n}$. Each $i$th action process only transitions when the state $w(k)$ transitions to an element of $\mathbb{W}_i$. At the time of transition, the probability that $a_i(k)$ becomes any $a_i \in \mathbb{A}_i$ is given by $\pi^i_{\theta_i}(a_i | w(k))$.

The state transitions depend on the prior state and action vector. In particular, let $P(w', a', w)$ be a transition kernel defining the probability of state $w$ given prior state $w'$ and action $a'$. Letting $\theta = (\theta_1, \ldots, \theta_n)$, we have

$$\Pr\{w(k) = w, a(k) = a | w(k-1) = w', a(k-1) = a', \theta\}$$
$$= P(w', a', w) \prod_{i: w \in \mathbb{W}_i} \pi^i_{\theta_i}(a_i | w) \prod_{i: w \notin \mathbb{W}_i} \mathbf{1}_{\{a'_i = a_i\}}.$$

Define $\mathbb{F}_k$ to be the $\sigma$-algebra generated by $\{(w(\ell), a(\ell)) | \ell = 1, \ldots, k\}$.

While the system is in state $w \in \mathbb{W}$ and action $a \in \mathbb{A}$ is applied, each component $i$ receives a reward $r_i(w, a)$. The average reward received by the network is $r(w, a) = \frac{1}{n} \sum_{i=1}^{n} r_i(w, a)$.

**Assumption 1.** *For every $\theta$, the Markov chain $w(k)$ is ergodic (aperiodic, irreducible).*

Given Assumption 1, for each fixed $\theta$, there is a well-defined long-term average reward $\lambda(\theta) = \lim_{K \to \infty} \frac{1}{K} \mathrm{E}[\sum_{k=0}^{K-1} r(w(k), a(k))]$.

We will consider a stochastic approximation iteration

$$(1) \qquad\qquad \theta_i(k+1) = \theta_i(k) + \epsilon \chi_i(k).$$

Here, $\epsilon > 0$ is a constant step size and $\chi_i(k)$ is a noisy estimate of the gradient $\nabla_{\theta_i} \lambda(\theta(k))$ computed at component $i$ based on the component's historically observed states, actions, and rewards, in addition to communication with other components. Our goal is to develop an estimator $\chi_i(k)$ that can be used in an adaptive, asynchronous, and decentralized context, and to establish the convergence of the resulting stochastic approximation scheme.

Our approach builds on policy gradient algorithms that have been proposed in recent years ([5, 7, 8, 3, 4, 2]). As a starting point, consider a gradient estimation method that is a decentralized variation of the OLPOMDP algorithm of [3, 4, 1]. In this algorithm, each component $i$ maintains and updates an eligibility vector $z_i^\beta(t) \in \mathbb{R}^{N_i}$, defined by

$$(2) \qquad\qquad z_i^\beta(k) = \sum_{\ell=0}^{k} \beta^{k-\ell} \frac{\nabla_{\theta_i} \pi^i_{\theta_i(\ell)}(a_i(\ell) | w(\ell))}{\pi^i_{\theta_i(\ell)}(a_i(\ell) | w(\ell))} \mathbf{1}_{\{w(\ell) \in \mathbb{W}_i\}},$$

for some $\beta \in (0, 1)$. The algorithm generates an estimate $\bar{\chi}_i(k) = r(w(t), a(t)) z_i^\beta(k)$ to the local gradient $\nabla_{\theta_i} \lambda(\theta(k))$. Note that while the credit vector $z_i^\beta(t)$ can be computed using only local information, the gradient estimate $\bar{\chi}_i(t)$ cannot be computed without knowledge of the global reward $r(x(t), a(t))$ at each time. In a fully decentralized environment, where components only have knowledge of their local rewards, this algorithm cannot be used.

In this paper, we present a simple scalable distributed protocol through which rewards occurring locally at each node are communicated over time across the network and gradient estimates are generated at each node based on local information. A fundamental issue this raises is that rewards may incur large delays before being communicated across the network. Moreover, these delays may be random and may correlated with the underlying events that occur in operation of the network. We address this issue and establish conditions for convergence. Another feature of the protocol is that it is completely decentralized – there is no central processor that aggregates and disseminates rewards. As such, the protocol is robust to isolated changes or failures in the network. In addition to design of the protocol, a significant contribution is in the protocol's analysis, which we believe requires new ideas beyond what has been employed in the prior policy gradient literature.

## 3 A General Framework for Protocols

We will make the following assumption regarding the policies, which is common in the policy gradient literature ([7, 8, 3, 4, 2]).

**Assumption 2.** *For all $i$ and every $w \in \mathbb{W}_i$, $a_i \in \mathbb{A}_i$, $\pi^i_{\theta_i}(a_i|w)$ is a continuously differentiable function of $\theta_i$. Further, for every $i$, there exists a bounded function $L_i(w, a_i, \theta)$ such that for all $w \in \mathbb{W}_i$, $a_i \in \mathbb{A}_i$, $\nabla_{\theta_i} \pi^i_{\theta_i}(a_i|w) = \pi^i_{\theta_i}(a_i|w) L_i(w, a_i, \theta)$.*

The latter part of the assumption is satisfied, for example, if there exists a constant $\epsilon > 0$ such that for each $i, w \in \mathbb{W}_i, a_i \in \mathbb{A}_i$, either $\pi^i_{\theta_i}(a_i|w) = 0$ for every $\theta_i$ or $\pi^i_{\theta_i}(a_i|w) \geq \epsilon$, for all $\theta_i$.

Consider the following gradient estimator:

$$(3) \qquad \chi_i(k) = z^\beta_i(k) \frac{1}{n} \sum_{j=1}^n \sum_{\ell=0}^k d^\alpha_{ij}(\ell, k) r_j(\ell),$$

where we use the shorthand $r_j(\ell) = r_j(w(\ell), a(\ell))$. Here, the random variables $\{d^\alpha_{ij}(\ell, k)\}$, with parameter $\alpha \in (0, 1)$, represent an arrival process describing the communication of rewards across the network. Indeed, $d^\alpha_{ij}(\ell, k)$ is the fraction of the reward $r_j(\ell)$ at component $j$ that is learned by component $i$ at time $k \geq \ell$. We will assume the arrival process satisfies the following conditions.

**Assumption 3.** *For each $i, j, \ell$, and $\alpha \in (0, 1)$, the process $\{d^\alpha_{ji}(\ell, k)|k = \ell, \ell+1, \ell+2, \ldots\}$ satisfies:*

1. *$d^\alpha_{ji}(\ell, k)$ is $\mathbb{F}_k$-measurable.*

2. *There exists a scalar $\gamma \in (0, 1)$ and a random variable $c_\ell$ such that for all $k \geq \ell$,*

$$\left| \frac{d^\alpha_{ji}(\ell, k)}{(1-\alpha)\alpha^{k-\ell}} - 1 \right| < c_\ell \gamma^{k-\ell},$$

   *with probability 1. Further, we require that the distribution of $c_\ell$ given $\mathbb{F}_\ell$ depend only on $(w(\ell), a(\ell))$, and that there exist a constant $\bar{c}$ such that $\mathrm{E}[c_\ell|w(\ell) = w, a(\ell) = a] < \bar{c} < \infty$, with probability 1 for all initial conditions $w \in \mathbb{W}$ and $a \in \mathbb{A}$.*

3. *The distribution of $\{d^\alpha_{ji}(\ell, k)|k = \ell, \ell+1, \ldots\}$ given $\mathbb{F}_\ell$ depends only on $w(\ell)$ and $a(\ell)$.*

The following result, proved in our appendix [9], establishes the convergence of the long-term sample averages of $\chi_i(t)$ of the form (3) to an estimate of the gradient. This type of convergence is central to the convergence of the stochastic approximation iteration (1).

**Theorem 1.** *Holding $\theta$ fixed, the limit*

$$\nabla_{\theta_i}^{\alpha\beta}\lambda(\theta) = \lim_{K \to \infty} \frac{1}{K} \mathrm{E}\left[\sum_{k=0}^{K-1} \chi_i(k)\right]$$

*exists. Further,*

$$\lim_{\alpha \uparrow 1} \limsup_{\beta \uparrow 1} \left\| \nabla_{\theta_i}^{\alpha\beta}\lambda(\theta) - \nabla_{\theta_i}\lambda(\theta) \right\| = 0.$$

.

## 4 Example: A Sensor Network

In this section, we present a model of a wireless network of sensors that gathers and communicates data to a central base station. Our example is motivated by issues arising in the development of sensor network technology being carried out by commercial producers of electronic devices. However, we will not take into account the many complexities associated with real sensor networks. Rather, our objective is to pose a simplified model that motivates and provides a context for discussion of our distributed optimization protocol.

### 4.1 System Description

Consider a network of $n$ sensors and a central base station. Each sensor gathers packets of data through observation of its environment, and these packets of data are relayed through the network to the base station via multi-hop wireless communication. Each sensor retains a queue of packets, each obtained either through sensing or via transmission from another sensor. Packets in a queue are indistinguishable – each is of equal size and must be transferred to the central base station. We take the state of a sensor to be the number of packets in the queue and denote the state of the $i$th sensor at time $k$ by $x_i(k)$. The number of packets in a queue cannot exceed a finite buffer size, which we denote by $\overline{x}$.

A number of triggering events occur at any given device. These include (1) packetizing of an observation (2) reception of a packet from another sensor, (3) transmission of a packet to another sensor, (4) awakening from a period of sleep, (5) termination of a period of attempted reception, (6) termination of a period of attempted transmission. At the time of a triggering event, the sensor must decide on its next action. Possible actions include (1) sleep, (2) attempt transmission, (3) attempt reception. When the buffer is full, options are limited to (1) and (2). When the buffer is empty, options are limited (1) and (3). The action taken by the $i$th sensor at time $k$ is denoted by $a_i(k)$.

The base station will be thought of as a sensor that has an infinite buffer and perpetually attempts reception. For each $i$, there is a set $\mathbb{N}(i)$ of entities with which the $i$th sensor can directly communicate. If the $i$th sensor is attempting transmission of a packet and there is at least one element of $\mathbb{N}(i)$ that is simultaneously attempting reception and is closer to the base station than component $i$, the packet is transferred to the queue of that element. If there are multiple such elements, one of them is chosen randomly. Note that if among the elements of $\mathbb{N}(i)$ that are attempting reception, all are further away from the base station than component $i$, no packet is transmitted.

Observations are made and packetized by each sensor at random times. If a sensor's buffer is not full when an observation is packetized, an element is added to the queue. Otherwise, the packet is dropped from the system.

### 4.2 Control Policies and Objective

Every sensor employs a control policy that selects an action based on its queue length each time a triggering event occurs. The action is maintained until occurrence of the next triggering event. Each $i$th sensor's control policy is parameterized by a vector $\theta_i \in \mathbb{R}^2$. Given $\theta_i$, at an event time, if the $i$th sensor has a non-empty queue, it chooses to transmit with probability $\theta_{i1}$. If the $i$th sensor does not transmit and its queue is not full, it chooses to receive with probability $\theta_{i2}$. If the sensor does not transmit or receive, then it sleeps. In order to satisfy Assumption 2, we constrain $\theta_{i1}$ and $\theta_{i2}$ to lie in an interval $[\theta_\ell, \theta_h]$, where $0 < \theta_\ell < \theta_h < 1$.

Assume that each sensor has a finite power supply. In order to guarantee a minimum lifespan for the network, we will require that each sensor sleeps at least a fraction $f_s$ of the time. This is enforced by considering a time window of length $T_s$. If, at any given time, a sensor has not slept for a total fraction of a least $f_s$ of the preceding time $T_s$, it is forced to sleep and hence not allowed to transmit or receive.

The objective is to minimize a weighted sum of the average delay and average number of dropped packets per unit of time. Delay can be thought of as the amount of time a packet spends in the network before arriving at the base station. Hence, the objective is:

$$\max_{\theta_1,\ldots,\theta_n} \limsup_{K\to\infty} -\frac{1}{K}\sum_{k=0}^{K-1}\frac{1}{n}\sum_{i=1}^{n}\left(x_i(k) + \xi D_i(k)\right),$$

where $D_i(k)$ is the number of packets dropped by sensor $i$ at time $k$, and $\xi$ is a weight reflecting the relative importance of delay and dropped packets.

## 5 Distributed Optimization Protocol

We now describe a simple protocol by which components a the network can communicate rewards, in a fashion that satisfies the requirements of Theorem 1 and hence will produce good gradient estimates. This protocol communicates the rewards across the network over time using a distributed averaging procedure.

In order to motivate our protocol, consider a different problem. Imagine each component $i$ in the network is given a real value $R_i$. Our goal is to design an asynchronous distributed protocol through which each node will obtain the average $\overline{R} = \sum_{i=1}^{n} R_i/n$. To do this, define the vector $Y(0) \in \mathbb{R}^n$ by $Y_i(0) = R_i$ for all $i$. For each edge $(i,j)$, define a function $Q^{(i,j)} : \mathbb{R}^n \mapsto \mathbb{R}^n$ by

$$Q_\ell^{(i,j)}(Y) = \begin{cases} \frac{Y_i+Y_j}{2} & \text{if } \ell \in \{i,j\}, \\ Y_\ell & \text{otherwise.} \end{cases}$$

At each time $t$, choose an edge $(i,j)$, and set $Y(k+1) = Q^{(i,j)}(Y(k))$. If the graph is connected and every edge is sampled infinitely often, then $\lim_{k\to\infty} Y(t) = \overline{Y}$, where $\overline{Y}_i = \overline{R}$. To see this, note that the operators $Q^{(i,j)}$ preserve the average value of the vector, hence $\sum_{i=1}^{n} Y_i(k)/n = \overline{R}$. Further, for any $k$, either $Y(k+1) = Y(k)$ or $\|Y(k+1)-\overline{Y}\| < \|Y(k) - \overline{Y}\|$. Further, $\overline{Y}$ is the unique vector with average value $\overline{R}$ that is a fixed point for all operators $Q^{(i,j)}$. Hence, as long as the graph is connected and each edge is sampled infinitely often, $Y_i(k)\to\overline{R}$ as $k\to\infty$ and the components agree to the common average $\overline{R}$.

In the context of our distributed optimization protocol, we will assume that each component $i$ maintains a scalar value $Y_i(k)$ at time $k$ representing an estimate of the global reward. We will define a structure by which components communicate. Define $E$ to be the set of edges along which communication can occur. For an ordered set of distinct edges $S =$

$\big((i_i, j_1), \dots, (i_{|S|}, j_{|S|})\big)$, define a set $\mathbb{W}_S \subset \mathbb{W}$. Let $\sigma(E)$ be the set of all possible ordered sets of disjoint edges $S$, including the empty set. We will assume that the sets $\{W_S | S \in \sigma(E)\}$ are disjoint and together form a partition of $\mathbb{W}$.

If $w(k) \in \mathbb{W}_S$, for some set $S$, we will assume that the components along the edges in $S$ communicate in the order specified by $S$. Define $Q^S = Q^{(i_{|S|}, j_{|S|})} \dots Q^{(i_1, j_1)}$, where the terms in the product are taken over the order specified by $S$. Define $R(k) = (r_1(k), \dots, r_n(k))$ is a vector of rewards occurring at time $k$. The update rule for the vector $Y(k)$ is given by $Y(k+1) = R(k+1) + \alpha Q^{S(k+1)} Y(k)$, where

$$Q^{S(k+1)} = \sum_{S \in \sigma(E)} \mathbf{1}_{\{w(k+1) \in \mathbb{W}_S\}} Q^S.$$

Let $\hat{E} = \{(i,j) | (i,j) \in S, \mathbb{W}_S \neq \emptyset\}$. We will make the following assumption.

**Assumption 4.** *The graph $(V, \hat{E})$ is connected.*

Since the process $(w(k), a(k))$ is aperiodic and irreducible (Assumption 1), this assumption guarantees that every edge on a connected subset of edges is sampled infinitely often.

Policy parameters are updated at each component according to the rule:

(4) $$\theta_i(k+1) = \theta_i(k) + \epsilon z_i^\beta(k)(1-\alpha) Y_i(k).$$

In relation to equations (1) and (3), we have

(5) $$d_{ji}^\alpha(\ell, k) = n(1-\alpha)\alpha^{k-\ell} \left[\hat{Q}(\ell, k)\right]_{ij},$$

where $\hat{Q}(\ell, k) = Q^{S(k-1)} \dots Q^{S(\ell)}$.

The following theorem, which relies on a general stochastic approximation result from [6] together with custom analysis available in our appendix [9], establishes the convergence of the distributed stochastic iteration method defined by (4).

**Theorem 2.** *For each $\epsilon > 0$, define $\{\theta^\epsilon(k) | k = 0, 1, \dots\}$ as the result of the stochastic approximation iteration (4) with the fixed value of $\epsilon$. Assume the set $\{\theta^\epsilon(k) | k, \epsilon\}$ is bounded. Define the continuous time interpolation $\bar{\theta}^\epsilon(t)$ by setting $\bar{\theta}^\epsilon(t) = \theta^\epsilon(k)$ for $t \in [k\epsilon, k\epsilon + \epsilon)$. Then, for any sequence of processes $\{\bar{\theta}^\epsilon(t) | \epsilon \to 0\}$ there exists a subsequence that weakly converges to $\bar{\theta}(t)$ as $\epsilon \to 0$, where $\bar{\theta}(t)$ is a solution to the ordinary differential equation*

(6) $$\dot{\bar{\theta}}(t) = \nabla_\theta^{\alpha\beta} \lambda(\bar{\theta}(t)).$$

*Further, define $\mathcal{L}$ to be the set of limit points of (6), and for a $\delta > 0$, $N_\delta(\mathcal{L})$ to be a neighborhood of radius $\delta$ about $\mathcal{L}$. The fraction of time that $\bar{\theta}^\epsilon(t)$ spends in $N_\delta(\mathcal{L})$ over the time interval $[0, T]$ goes to 1 in probability as $\epsilon \to 0$ and $T \to \infty$.*

Note that since we are using a constant step-size $\epsilon$, this type of weak convergence is the strongest one would expect. The parameters will typically oscillate in the neighborhood of an limit point, and only weak convergence to a distribution centered around a limit point can be established. An alternative would be to use a decreasing step size $\epsilon(k) \to 0$ in (4). In such instances, probability 1 convergence to a local optimum can often be established. However, with decreasing step sizes, the adaptation of parameters becomes very slow as $\epsilon(n)$ decays. We expect our protocol to be used in an online fashion, where it is ideal to be adaptive to long-term changes in network topology or dynamics of the environment. Hence, the constant step size case is more appropriate as it provides such adaptivity.

Also, a boundedness requirement on the iterates in Theorem 2 is necessary for the mathematical analysis of convergence. In practical numerical implementations, choices of the policy parameters $\theta_i$ would be constrained to bounded sets of $H_i \subset \mathbb{R}^{N_i}$. In such an implementation, the iteration (4) would be replaced with an iteration projected onto the set $H_i$. The conclusions of Theorem 2 would continue to hold, but with the ODE (6) replaced with an appropriate projected ODE. See [6] for further discussion.

### 5.1 Relation to the Example

In the example of Section 4, one approach to implementing our distributed optimization protocol involves passing messages associated with the optimization protocol alongside normal network traffic, as we will now explain. Each $i$th sensor should maintain and update two vectors: a parameter vector $\theta_i(k) \in \mathbb{R}^2$ and an eligibility vector $z_i^\beta(k)$. If a triggering event occurs at sensor $i$ at time $k$, the eligibility vector is updated according to

$$z_i^\beta(k) = \beta z^\beta(k-1) + \frac{\nabla_{\theta_i} \pi_{\theta_i(k)}^i (a_i(k)|w(k))}{\pi_{\theta_i(k)}^i (a_i(k)|w(k))}.$$

Otherwise, $z_i^\beta(k) = \beta z_i^\beta(k-1)$. Furthermore, each sensor maintains an estimate $Y_i(k)$ of the global reward. At each time $k$, each $i$th sensor observes a reward (negative cost) of $r_i(k) = -x_i(k) - \xi D_i(k)$. If two neighboring sensors are both not asleep at a time $k$, they communicate their global reward estimates from the previous time. If the $i$th sensor is not involved in a reward communication event at that time, its global reward estimate is updated according to $Y_i(k) = \alpha Y_i(k-1) + r_i(k)$. On the other hand, at any time $k$ that there is a communication event, its global reward estimate is updated according to $Y_i(k) = r_i(k) + \alpha(Y_i(k) + \alpha Y_j(k))/2$, where $j$ is the index of the sensor with which communication occurs. If communication occurs with multiple neighbors, the corresponding global reward estimates are averaged pairwise in an arbitrary order. Clearly this update process can be modeled in terms of the sets $\mathbb{W}_S$ introduced in the previous section. In this context, the graph $\hat{E}$ contains an edge for each pair of neighbors in the sensor network, where the neighborhood relations are capture by $\mathbb{N}$, as introduced in Section 4. To optimize performance over time, each $i$th sensor would update its parameter values according to our stochastic approximation iteration (4).

To highlight the simplicity of this protocol, note that each sensor need only maintain and update a few numerical values. Furthermore, the only communication required by the optimization protocol is that an extra scalar numerical value be transmitted and an extra scalar numerical value be received during the reception or transmission of any packet.

As a numerical example, consider the network topology in Figure 1. Here, at every time step, an observation arrives at a sensor with a $0.02$ probability, and each sensor maintains a queue of up to 20 observations. Policy parameters $\theta_{i1}$ and $\theta_{i2}$ for each sensor $i$ are constrained to lie in the interval $[0.05, 0.95]$. (Note that for this set of parameters, the chance of a buffer overflow is very small, and hence did not occur in our simulations.) A baseline policy is defined by having leaf nodes transmit with maximum probability, and interior nodes splitting their time roughly evenly between transmission and reception, when not forced to sleep by the power constraint.

Applying our decentralized optimization method to this example, it is clear in Figure 2 that the performance of the network is quickly and dramatically improved. Over time, the algorithm converges to the neighborhood of a local optimum as expected. Further, the algorithm achieves qualitatively similar performance to gradient optimization using the centralized OLPOMDP method of [3, 4, 1], hence decentralization comes at no cost.

## 6 Remarks and Further Issues

We are encouraged by the simplicity and scalability of the distributed optimization protocol we have presented. We believe that this protocol represents both an interesting direction for practical applications involving networks of electronic devices and a significant step in the policy gradient literature. However, there is an important outstanding issue that needs to be addressed to assess the potential of this approach: whether or not parameters can be adapted fast enough for this protocol to be useful in applications. There are two dimensions

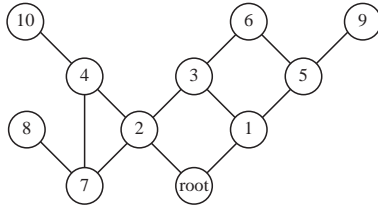

Figure 1: Example network topology.

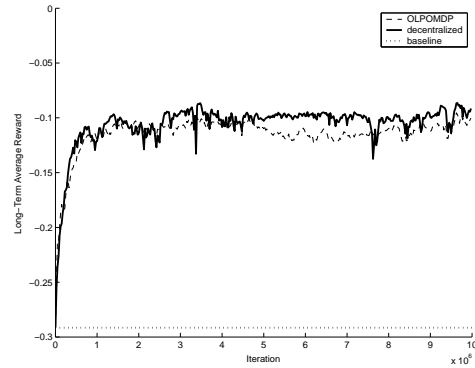

Figure 2: Convergence of method.

to this issue: (1) variance of gradient estimates and (2) convergence rate of the underlying ODE. Both should be explored through experimentation with models that capture practical contexts. Also, there is room for research that explores how variance can be reduced and the convergence rate of the ODE can be accelerated.

## Acknowledgements

The authors thank Abbas El Gamal, Abtin Keshavarzian, Balaji Prabhakar, and Elif Uysal for stimulating conversations on sensor network models and applications. This research was supported by NSF CAREER Grant ECS-9985229 and by the ONR under grant MURI-N00014-00-1-0637. The first author was also supported by a Benchmark Stanford Graduate Fellowship.

## References

[1] P. L. Bartlett and J. Baxter. Stochastic Optimization of Controlled Markov Decision Processes. In *IEEE Conference on Decision and Control*, pages 124–129, 2000.

[2] P. L. Bartlett and J. Baxter. Estimation and Approximation Bounds for Gradient-Based Reinforcement Learning. *Journal of Computer and System Sciences*, 64:133–150, 2002.

[3] J. Baxter and P. L. Bartlett. Infinite-Horizon Gradient-Based Policy Search. *Journal of Artificial Intelligence Research*, 15:319–350, 2001.

[4] J. Baxter, P. L. Bartlett, and L. Weaver. Infinite-Horizon Gradient-Based Policy Search: II. Gradient Ascent Algorithms and Experiments. *Journal of Artificial Intelligence Research*, 15:351–381, 2001.

[5] T. Jaakkola, S. P. Singh, and M. I. Jordan. Reinforcement Learning Algorithms for Partially Observable Markov Decision Problems. In *Advances in Neural Information Processing Systems 7*, pages 345–352, 1995.

[6] H. J. Kushner and G. Yin. *Stochastic Approximation Algorithms and Applications*. Springer-Verlag, New York, NY, 1997.

[7] P. Marbach, O. Mihatsch, and J.N. Tsitsiklis. Call Admission Control and Routing in Integrated Service Networks. In *IEEE Conference on Decision and Control*, 1998.

[8] P. Marbach and J.N. Tsitsiklis. Simulation–Based Optimization of Markov Reward Processes. *IEEE Transactions on Automatic Control*, 46(2):191–209, 2001.

[9] C. C. Moallemi and B. Van Roy. Appendix to NIPS Submission. URL: `http://www.moallemi.com/ciamac/papers/nips-2003-appendix.pdf`, 2003.
